# A Connectionist Symbol Manipulator That Discovers the Structure of Context-Free Languages

**Michael C. Mozer and Sreerupa Das**
Department of Computer Science &
Institute of Cognitive Science
University of Colorado
Boulder, CO 80309-0430

## Abstract

We present a neural net architecture that can discover hierarchical and recursive structure in symbol strings. To detect structure at multiple levels, the architecture has the capability of reducing symbols substrings to single symbols, and makes use of an external stack memory. In terms of formal languages, the architecture can learn to parse strings in an LR(0) context-free grammar. Given training sets of positive and negative exemplars, the architecture has been trained to recognize many different grammars. The architecture has only one layer of modifiable weights, allowing for a straightforward interpretation of its behavior.

Many cognitive domains involve complex sequences that contain hierarchical or recursive structure, e.g., music, natural language parsing, event perception. To illustrate, "the spider that ate the hairy fly" is a noun phrase containing the embedded noun phrase "the hairy fly." Understanding such multilevel structures requires forming *reduced descriptions* (Hinton, 1988) in which a string of symbols or states ("the hairy fly") is reduced to a single symbolic entity (a noun phrase). We present a neural net architecture that learns to encode the structure of symbol strings via such reduction transformations.

The difficult problem of extracting multilevel structure from complex, extended sequences has been studied by Mozer (1992), Ring (1993), Rohwer (1990), and Schmidhuber (1992), among others. While these previous efforts have made some

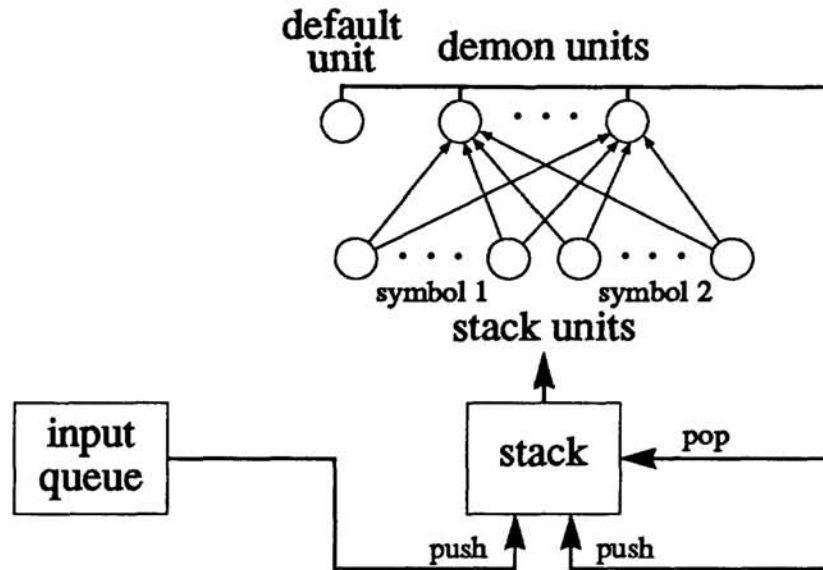

Figure 1: The demon model.

progress, no one has claimed victory over the problem. Our approach is based on a new perspective—one of symbolic reduction transformations—which affords a fresh attack on the problem.

# 1   A BLACKBOARD ARCHITECTURE

Our inspiration is a blackboard style architecture that works as follows. The input, a sequence of symbols, is copied onto a *blackboard*—a scratch pad memory—one symbol at a time. A set of *demons* watch over the blackboard, each looking for a specific pattern of symbols. When a demon observes its pattern, it *fires*, causing the pattern to be replaced by a symbol associated with that demon, which we'll call its *identity*. This process continues until the entire input string has been read or no demon can fire. The sequence of demon firings and the final blackboard contents specify the structure of the input.

The model we present is a simplified version of this blackboard architecture. The blackboard is implemented as a stack. Consequently, the demons have no control over *where* they write or read a symbol; they simply push and pop symbols from the stack. The other simplification is that the demon firing is based on template matching, rather than a more sophisticated form of pattern matching.

The demon model is sketched in Figure 1. An *input queue* holds the input string to be parsed, which is gradually transferred to the stack. The top $k$ stack symbols are encoded in a set of *stack units*; in the current implementation, $k = 2$. Each demon is embodied by a special processing unit which receives input from the stack units. The weights of each *demon unit* specify a pair of symbols, which the demon unit matches against the two stack symbols. If there is a match, the demon unit pops the top two stack symbols and pushes its identity. If no demon unit matches, an additional unit, called the *default unit*, becomes active. The default unit is responsible for transferring a symbol from the input queue onto the stack.

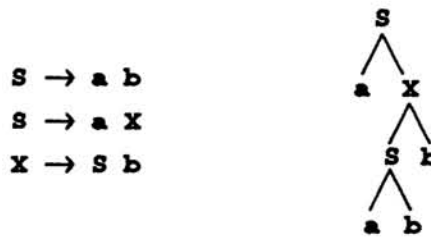

$$
\begin{aligned}
S &\rightarrow a\ b \\
S &\rightarrow a\ X \\
X &\rightarrow S\ b
\end{aligned}
$$

Figure 2: The rewrite rules defining a grammar that generates strings of the form $a^n b^n$ and a parse tree for the string aabb.

## 2   PARSING CONTEXT-FREE LANGUAGES

Each demon unit reduces a pair of symbols to a single symbol. We can express the operation of a demon as a rewrite rule of the form $X \rightarrow a\ b$, where the lower case letters denote symbols in the input string and upper case letters denote the demon identities, also symbols in their own right. The above rule specifies that when the symbols a and b appear on the top of the stack, in that order, the $X$ demon unit should fire, erasing those two symbols and replacing them with an $X$. Demon units can respond to internal symbols (demon identities) instead of input symbols, allowing internal symbols on the right hand side of the rule. Demon units can also respond to individual input symbols, achieving rules of the form $X \rightarrow a$.

Multiple demon units can have the same identity, leading to rewrite rules of a more general form, e.g., $X \rightarrow a\ b \mid Y\ c \mid d\ Z \mid a$. This class of rewrite rules can express a subset of context-free grammars. Figure 2 shows a sample grammar that generates strings of the form $a^n b^n$ and a parse tree for the input string aabb. The demon model essentially constructs such parse trees via the sequence of reduction operations.

That each rule has only one or two symbols on the right hand side imposes no limitation on the class of grammars that can be recognized. However, the demon model does require certain knowledge about the grammars to be identified. First, the maximum number of rewrite rules and the maximum number of rules having the same left-hand side must be specified in advance. This is because the units have to be allocated prior to learning. Second, the LR-class of the grammar must be given. To explain, any context-free grammar can be characterized as LR($n$), which indicates that the strings of the grammar can be parsed from left to right with $n$ symbols of look ahead on the input queue. The demon model requires that $n$ be specified in advance. In the present work, we examine only LR(0) grammars, but the architecture can readily be generalized to arbitrary $n$.

Giles et al. (1990), Sun et al. (1990), and Das, Giles, and Sun (1992) have previously explored the learning of context-free grammars in a neural net. Their approach was based on the automaton perspective of a recognizer, where the primary interest was to learn the dynamics of a pushdown automaton. There has also been significant work in context-free grammar inference using symbolic approaches. In general, these approaches require a significant amount of prior information about the grammar and, although theoretically sound, have not proven terribly useful in practice. A promising exception is the recent proposal of Stolcke (1993).

## 3    CONTINUOUS DYNAMICS

So far, we have described the model in a discrete way: demon firing is all-or-none and mutually exclusive, corresponding to the demon units achieving a unary representation. This may be the desired behavior following learning, but neural net learning algorithms like back propagation require exploration in continuous state and weight spaces and therefore need to allow partial activity of demon units. The continuous activation dynamics follow.

Demon unit $i$ computes the distance between its weights, $\mathbf{w}_i$, and the input, $\mathbf{x}$: $dist_i = b_i|\mathbf{w}_i - \mathbf{x}|^2$, where $b_i$ is an adjustable bias associated with the unit. The activity of unit $i$, denoted $s_i$, is computed via a normalized exponential transform (Bridle, 1990; Rumelhart, in press),

$$s_i = \frac{e^{-dist_i}}{\sum_j e^{-dist_j}},$$

which enforces a competition among the units. A special unit, called the default unit, is designed to respond when none of the demons fire strongly. Its activity, $s_{def}$, is computed like that of any demon unit with $dist_{def} = b_{def}$.

## 4    CONTINUOUS STACK

Because demon units can be partially active, stack operations need to be performed partially. This can be accomplished with a *continuous stack* (Giles et al., 1990). Unlike a discrete stack where an item is either present or absent, items can be present to varying degrees. Each item on the stack has an associated *thickness*, a scalar in the interval $[0, 1]$ indicating what fraction of the item is present (Figure 3).

To understand how the thickness plays a role in processing, we digress briefly and explain the encoding of symbols. Both on the stack and in the network, symbols are represented by numerical vectors that have one component per symbol. The vector representation of some symbol $\mathbf{X}$, denoted $\mathbf{r_X}$, has value 1 for the component corresponding to $\mathbf{X}$ and 0 for all other components. If the symbol has thickness $t$, the vector representation is $t\mathbf{r_X}$.

Although items on the stack have different thicknesses, the network is presented with *composite symbols* having thickness 1.0. Composite symbols are formed by combining stack items. For example, in Figure 3, composite symbol 1 is defined as the vector $.2\mathbf{r_X} + .5\mathbf{r_Z} + .3\mathbf{r_Y}$. The input to the demon network consists of the top two composite symbols on the stack.

The advantages of a continuous stack are twofold. First, it is required for network learning; if a discrete stack were used, a small change in weights could result in a big (discrete) change in the stack. This was the motivation underlying the continuous stack used by Giles et al. Second, the continuous stack is differentiable and hence allows us to back propagate error through the stack during learning. While we have summarized this point in one sentence, the reader must appreciate the fact that it is no small feat! Giles et al. did not consider back propagation through the stack.

Each time step, the network performs two operations on the stack:

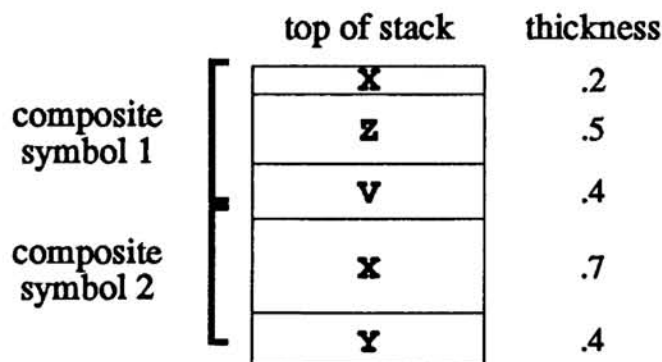

**Figure 3:** A continuous stack. The symbols indicate the contents; the height of a stack entry indicates its thickness, also given by the number to the right. The top composite symbol on the stack is a combination of the items forming a total thickness of 1.0; the next composite symbol is a combination of the items making up the next 1.0 units of thickness.

**Pop.** If a demon unit fires, the top two composite symbols should be popped from the stack (to be replaced by the demon's identity). If no demon unit fires, in which case the default unit becomes active, the stack should remain unchanged. These behaviors, as well as interpolated behaviors, are achieved by multiplying by $s_{def}$ the thickness of any portion of a stack item contributing to the top two composite symbols. Remember that $s_{def}$ is 0 when one or more demon units are strongly active, and is 1 when the default unit is fully active.

**Push.** The symbol written onto the stack is the composite symbol formed by summing the identity vectors of the demon units, weighted by their activities: $\sum_i s_i \mathbf{r}_i$, where $\mathbf{r}_i$ is the vector representing demon $i$'s identity. Included in this summation is the default unit, where $\mathbf{r}_{def}$ is defined to be the composite symbol over thickness $s_{def}$ of the input queue. (After a thickness of $s_{def}$ is read from the input queue, it is removed from the queue.)

## 5  TRAINING METHODOLOGY

The system is trained on positive and negative examples of a context-free grammar. Its task is to classify each input string as grammatical or not. Because the grammars can always be written such that the root of the parse tree is the symbol S (e.g., Figure 2), the stack should contain just S upon completion of processing of a positive example. For a negative example, the stack should contain anything but S.

These criteria can be translated into an objective function as follows. If one assumes a Gaussian noise distribution over outputs, the probability that the top of the stack contains the symbol S following presentation of example $i$ is

$$p_i^{root} \propto e^{-|\mathbf{c}_i - \mathbf{r}_S|^2},$$

where $\mathbf{c}_i$ is the vector representing the top composite symbol on the stack; and the probability that the total thickness of the stack is 1 (i.e., the stack contains exactly one item) is

$$p_i^{thick} \propto e^{-\Phi(T_i-1)^2},$$

where $T_i$ is the total thickness of the stack and $\Phi$ is a constant. For a positive example, the objective function should be greatest when there is a high probability of S being on the stack and a high probability of it being the sole item on the stack; for a negative example, the objective function should be greatest when either event has a low probability. We thus obtain a likelihood objective function whose logarithm the learning procedure attempts to maximize:

$$L = \prod_{i \in \text{pos example}} p_i^{root} p_i^{thick} \prod_{i \in \text{neg example}} (1 - p_i^{root} p_i^{thick}).$$

Training sets were generated by hand, with a preference for shorter strings. Positive examples were generated from the grammar; negative examples were either randomly generated or were formed by perturbing a grammatical string. In most training sets, there were roughly 3-5 times as many negative examples as positive. One might validly be concerned that we introduced some bias in our selection of examples. If so, it was not deliberate. In the initial experiments reported below, our goal was primarily to demonstrate that under some conditions, the network could actually induce the grammar. In the next phase of our research, we plan a systematic investigation of the number and nature of examples required for successful learning.

The total number of demon units and the (fixed) identity of each was specified in advance of learning. For the grammar in Figure 2, we provided at least two S demons and one X demon. Any number of demons beyond the minimum did not affect performance. The initial weights $\{w_{ij}\}$ were selected from a uniform distribution over the interval [.45, .55]. The $b_i$ were initialized to 1.0.

Before an example is presented, the stack is reset to contain only a single symbol, the null symbol with vector representation 0 and infinite thickness. The example string is placed in the input queue. The network is then allowed to run for $2l-1$ time steps, which is exactly the number of steps required to process any grammatical string of length $l$. One can intuit this fact by considering that it takes two operations to process each symbol, one to transfer the symbol from the input queue to the stack, and another to reduce the symbol.

The derivative of the objective function is computed with respect to the weight parameters using a form of back propagation through time (Rumelhart, Hinton, & Williams, 1986). This involves "unfolding" the architecture in time and back propagating through the stack. Weights are then updated to perform gradient ascent in the log likelihood function.

## 6  RESULTS AND DISCUSSION

We have successfully trained the architecture on a variety of grammars, including those shown in Table 1. In each case, the network discriminates positive and negative examples perfectly on the training set. For the first three grammars, additional (longer) strings were used to test network generalization performance. In each case, generalization performance was 100%.

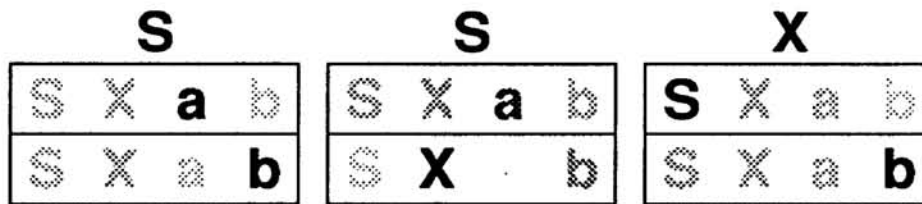

Figure 4: Sample weights for $a^n b^n$. Weights are organized by demon unit, whose identities appear above the rectangles. The top and bottom halves of the rectangle represents connections from composite symbols 1 and 2, respectively. The darker the shading is of a symbol in a rectangle, the larger the connection strength is from the input unit representing that symbol to the demon unit. The weights clearly indicate the three rewrite rules of the grammar.

Table 1: Grammars successfully learned by the demon model

| grammar name | rewrite rules |
|---|---|
| $a^n b^n$ | $S \to a\,b \mid a\,X$ <br> $X \to S\,b$ |
| parenthesis balancing | $S \to (\,) \mid (\,X \mid S\,S$ <br> $X \to S\,)$ |
| postfix | $S \to Y\,X \mid S\,X$ <br> $X \to Y + \mid S +$ <br> $Y \to a \mid b$ |
| pseudo natural language | $S \to NP\,VP$ <br> $NP \to d\,NP_2 \mid NP_2$ <br> $NP_2 \to n \mid a\,n$ <br> $VP \to v\,NP$ |

Due to the simplicity of the architecture—the fact that there is only one layer of modifiable weights—the learned weights can often be interpreted as symbolic rewrite rules (Figure 4). It is a remarkable achievement that the numerical optimization framework of neural net learning can be used to discover symbolic rules (see also Mozer & Bachrach, 1991).

The first three grammars were successfully learned by the model of Giles et al. (1990), although the analysis required to interpret the weights is generally more cumbersome and tentative. The last grammar could not be learned by their model (Das et al., 1992).

When more demon units are provided to the model than are required for the domain, the weights tend to be less interpretable, but generalization performance is just as good. (Of course, this result can hold for only a limited range of network sizes.) The model also does well with very small training sets (e.g., three positive, three negative examples for $a^n b^n$). This is no doubt because the architecture imposes strong biases on the learning process. We performed some preliminary experiments with staged training in which the length of strings in the training set was increased gradually, allowing the model to first learn simple cases and then move on to more difficult cases. This substantially improved the training time and robustness.

Although the current version of the model is designed for LR(0) context-free grammars, it can be extended to LR($n$) by including connections from the first $n$ composite symbols in the input queue to the demon units. However, our focus is not necessarily on building the theoretically most powerful formal language recognizer and learning system; rather, our primary interest has been on integrating symbol manipulation capabilities into a neural network architecture. In this regard, the model makes a clear contribution. It has the ability represent a string of symbols with a single symbol, and to do so iteratively, allowing for the formation of hierarchical and recursive structures. This is the essence of symbolic information processing, and, in our view, a key ingredient necessary for structure learning.

## Acknowledgements

This research was supported by NSF Presidential Young Investigator award IRI–9058450 and grant 90–21 from the James S. McDonnell Foundation. Our thanks to Paul Smolensky, Lee Giles, and Jürgen Schmidhuber for helpful comments regarding this work.

## References

Bridle, J. (1990). Training stochastic model recognition algorithms as networks can lead to maximum mutual information estimation of parameters. In D. S. Touretzky (Ed.), *Advances in neural information processing systems 2* (pp. 211–217). San Mateo, CA: Morgan Kaufmann.

Das, S., Giles, C. L., & Sun, G. Z. (1992). Learning context-free grammars: Capabilities and limitations of neural network with an external stack memory. In *Proceedings of the Fourteenth Annual Conference of the Cognitive Science* (pp. 791–795). Hillsdale, NJ: Erlbaum.

Giles, C. L., Sun, G. Z., Chen, H. H., Lee, Y. C., & Chen, D. (1990). Higher order recurrent networks and grammatical inference. In D. S. Touretzky (Ed.), *Advances in neural information processing systems 2* (pp. 380–387). San Mateo, CA: Morgan Kaufmann.

Hinton, G. E. (1988). Representing part–whole hierarchies in connectionist networks. *Proceedings of the Eighth Annual Conference of the Cognitive Science Society.*

Mozer, M. C. (1992). The induction of multiscale temporal structure. In J. E. Moody, S. J. Hanson, & R. P. Lippman (Eds.), *Advances in neural information processing systems IV* (pp. 275–282). San Mateo, CA: Morgan Kaufmann.

Mozer, M. C., & Bachrach, J. (1991). SLUG: A connectionist architecture for inferring the structure of finite-state environments. *Machine Learning, 7,* 139–160.

Ring, M. (1993). Learning sequential tasks by incrementally adding higher orders. *This volume.*

Rohwer, R. (1990). The 'moving targets' training algorithm. In D. S. Touretzky (Ed.), *Advances in neural information processing systems 2* (pp. 558–565). San Mateo, CA: Morgan Kaufmann.

Rumelhart, D. E., Hinton, G. E., & Williams, R. J. (1986). Learning internal representations by error propagation. In D. E. Rumelhart & J. L. McClelland (Eds.), *Parallel distributed processing: Explorations in the microstructure of cognition. Volume I: Foundations* (pp. 318–362). Cambridge, MA: MIT Press/Bradford Books.

Rumelhart, D. E. (in press). Connectionist processing and learning as statistical inference. In Y. Chauvin & D. E. Rumelhart (Eds.), *Backpropagation: Theory, architectures, and applications.* Hillsdale, NJ: Erlbaum.

Schmidhuber, J. (1992). Learning unambiguous reduced sequence descriptions. In J. E. Moody, S. J. Hanson, & R. P. Lippman (Eds.), *Advances in neural information processing systems IV* (pp. 291–298). San Mateo, CA: Morgan Kaufmann.

Stolcke, A., & Omohundro, S. (1993). Hidden markov model induction by Bayesian model merging. *This volume.*

Sun, G. Z., Chen, H. H., Giles, C. L., Lee, Y. C., & Chen, D. (1990). Connectionist pushdown automata that learn context-free grammars. In *Proceedings of the International Joint Conference on Neural Networks* (pp. I-577). Hillsdale, NJ: Erlbaum Associates.